# Decontaminating Human Judgments by Removing Sequential Dependencies

**Michael C. Mozer,**[*] **Harold Pashler,**[†] **Matthew Wilder,**[*]
**Robert V. Lindsey,**[*] **Matt C. Jones,**[°] **& Michael N. Jones**[‡]
[*] Dept. of Computer Science, University of Colorado
[†] Dept. of Psychology, UCSD
[°] Dept. of Psychology, University of Colorado
[‡] Dept. of Psychological and Brain Sciences, Indiana University

## Abstract

For over half a century, psychologists have been struck by how poor people are at expressing their internal sensations, impressions, and evaluations via rating scales. When individuals make judgments, they are incapable of using an absolute rating scale, and instead rely on reference points from recent experience. This *relativity of judgment* limits the usefulness of responses provided by individuals to surveys, questionnaires, and evaluation forms. Fortunately, the cognitive processes that transform internal states to responses are not simply noisy, but rather are influenced by recent experience in a lawful manner. We explore techniques to remove sequential dependencies, and thereby decontaminate a series of ratings to obtain more meaningful human judgments. In our formulation, decontamination is fundamentally a problem of inferring latent states (internal sensations) which, because of the relativity of judgment, have temporal dependencies. We propose a decontamination solution using a conditional random field with constraints motivated by psychological theories of relative judgment. Our exploration of decontamination models is supported by two experiments we conducted to obtain ground-truth rating data on a simple length estimation task. Our decontamination techniques yield an over 20% reduction in the error of human judgments.

## 1   Introduction

Suppose you are asked to make a series of moral judgments by rating, on a 1–10 scale, various actions, with a rating of 1 indicating 'not particularly bad or wrong' and a rating of 10 indicating 'extremely evil.' Consider the series of actions on the left.

(1) Stealing a towel from a hotel      (1′) Testifying falsely for pay
(2) Keeping a dime you find on the ground      (2′) Using guns on striking workers
(3) Poisoning a barking dog      (3′) Poisoning a barking dog

Now consider that instead you had been shown the series on the right. Even though individuals are asked to make absolute judgments, the mean rating of statement (3) in the first context is reliably higher than the mean rating of the identical statement (3′) in the second context (Parducci, 1968). The classic explanation of this phenomenon is cast in terms of *anchoring* or *primacy*: information presented early in time serves as a basis for making judgments later in time (Tversky & Kahneman, 1974). In the Netflix contest, significant attention was paid to anchoring effects by considering that an individual who gives high ratings early in a session is likely to be biased toward higher ratings later in a session (Koren, August 2009; Ellenberg, March 2008).

The need for anchors comes from the fact that individuals are poor at or incapable of making absolute judgments and instead must rely on reference points to make relative judgments (e.g., Laming, 1984; Parducci, 1965, 1968; Stewart, Brown, & Chater, 2005). Where do these reference points come from? There is a rich literature in experimental and theoretical psychology exploring *sequential*

*dependencies* suggesting that reference points change from one trial to the next in a systematic manner. (We use the psychological jargon 'trial' to refer to a single judgment or rating in a series.)

Sequential dependencies occur in many common tasks in which an individual is asked to make a series of responses, such as filling out surveys, questionnaires, and evaluations (e.g., usability ratings, pain assessment inventories). Every faculty member is aware of drift in grading that necessitates comparing papers graded early on a stack with those graded later. Recency effects have been demonstrated in domains as varied as legal reasoning and jury evidence interpretation (Furnham, 1986; Hogarth & Einhorn, 1992) and clinical assessments (Mumma & Wilson, 2006).

However, the most carefully controlled laboratory studies of sequential dependencies, dating back to the the 1950's (discussed by Miller, 1956), involve the rating of unidimensional stimuli, such as the loudness of a tone or the length of a line. Human performance at rating stimuli is surprisingly poor compared to an individual's ability to discriminate the same stimuli. Regardless of the domain, responses convey not much more than 2 bits of mutual information with the stimulus (Stewart et al., 2005). Different types of judgment tasks have been studied including *absolute identification*, in which the individual's task is to specify the distinct stimulus level (e.g., 10 levels of loudness), *magnitude estimation*, in which the task is to estimate the magnitude of a stimulus which may vary continuously along a dimension, and *categorization* which is a hybrid task requiring individuals to label stimuli by range. Because the number of responses in absolute identification and categorization tasks is often quite large, and because individuals are often not aware of the discreteness of stimuli in absolute identification tasks, there isn't a qualitative difference among tasks. Feedback is typically provided, especially in absolute identification and categorization tasks. Without feedback, there are no explicit anchors against which stimuli can be assessed.

The pattern of sequential effects observed is complex. Typically, experimental trial $t$, trial $t-1$ has a large influence on ratings, and trials $t-2$, $t-3$, etc., have successively diminishing influences. The influence of recent trials is exerted by *both* the stimuli and responses, a fact which makes sense in light of the assumption that individuals form their response on the current trial by analogy to recent trials (i.e., they determine a response to the current stimulus that has the same relationship as the previous response had to the previous stimulus). Both *assimilation* and *contrast* effects occur: an assimilative response on trial $t$ occurs when the response moves in the direction of the stimulus or response on trial $t-k$; a contrastive response is one that moves away. Interpreting recency effects in terms of assimilation and contrast is nontrivial and theory dependent (DeCarlo & Cross, 1990).

Many mathematical models have been developed to explain the phenomena of sequential effects in judgment tasks. All adopt the assumption that the transduction of a stimulus to its internal representation is veridical. We refer to this internal representation as the *sensation*, as distinguished from the external stimulus. (For judgments of nonphysical quantities such as emotional states and affinities, perhaps the terms *impression* or *evaluation* would be more appropriate than sensation.) Sequential dependencies and other corruptions of the representation occur in the mapping of the sensation to a response. According to all theories, this mapping requires reference to previous sensation-response pairings. However, the theories differ with respect to the reference set. At one extreme, the theory of Stewart et al. (2005) assumes that only the previous sensation-response pair matters. Other theories assume that multiple sensation-response anchors are required, one fixed and unchanging and another varying from trial to trial (e.g., DeCarlo & Cross, 1990). And in categorization and absolute identification tasks, some theories posit anchors for each distinct response, which are adjusted trial-to-trial (e.g., Petrov & Anderson, 2005). Range-frequency theory (Parducci, 1965) claims that sequential effects arise because the sensation-response mapping is adjusted to utilize the full response range, and to produce roughly an equal number of responses of each type. This effect is the consequence of many other theories, either explicitly or implicitly.

Because recent history interacts with the current stimulus to determine an individual's response, responses have a complex relationship with the underlying sensation, and do not provide as much information about the internal state of the individual as one would hope. In the applied psychology literature, awareness of sequential dependencies has led some researchers to explore strategies that mitigate relativity of judgment, such as increasing the number of response categories and varying the type and frequency of anchors (Mumma & Wilson, 2006; Wedell, Parducci, & Lane, 1990).

In contrast, our approach to extracting more information from human judgments is to develop automatic techniques that recover the underlying sensation from a response that has been contaminated

by cognitive processes producing the response. We term this recovery process *decontamination*. As we mentioned earlier, there is some precedent in the Netflix competition for developing empirical approaches to decontamination. However, to the best of our knowledge, the competitors were not focused on trial-to-trial effects, and their investigation was not systematic. Systematic investigation requires ground truth knowledge of the individuals' sensations.

## 2   Experiments

To collect ground-truth data for use in the design of decontamination techniques, we conducted two behavioral experiments using stimuli whose magnitudes could be objectively determined. In both experiments, participants were asked to judge the horizontal gap between two vertically aligned dots on a computer monitor. The position of the dots on the monitor shifted randomly from trial to trial. Participants were asked to respond to each dot pair using a 10-point rating scale, with 1 corresponding to the smallest gap they would see, and 10 corresponding to the largest.

The task requires absolute identification of 10 distinct gaps. The participants were only told that their task was to judge the distance between the dots. They were not told that only 10 unique stimuli were presented, and were likely unaware of this fact (memory of exact absolute gaps is too poor), and thus the task is indistinguishable from a magnitude estimation or categorization task in which the gap varied continuously. The experiment began with a practice block of ten trials. During the practice block, participants were shown every one of the ten gaps in random order, and simultaneous with the stimulus they were told—via text on the screen below the dots—the correct classification. After the practice blocks, no further feedback was provided. Although the psychology literature is replete with line-length judgment studies (two recent examples: Lacouture, 1997; Petrov & Anderson, 2005), the vast majority provide feedback to participants on at least some trials beyond the practice block. We wanted to avoid the anchoring provided by feedback in order that the task is more analogous to the the type of survey tasks we wish to decontaminate, e.g., the Netflix movie scores. Another distinction between our experiments and previous experiments is an attempt to carefully control the sequence structure, as described next.

### 2.1   Experiment Methodology

In Experiment 1, the practice block was followed by 2 blocks of 90 trials. Within a block, the trial sequence was arranged such that each gap was preceded exactly once by each other gap, with the exception that no repetitions occurred. Further, every ten trials in a block consisted of exactly one presentation of each gap. In Experiment 2, the practice block was followed by 2 blocks of 100 trials. The constraint on the sequence in Experiment 2 was looser than in Experiment 1: within a block, each gap occurred exactly once preceded by each other gap. However, repetitions were included, and there was no constraint on the subblocks of ten trials. The other key difference between experiments was the gap lengths. In Experiment 1, gap $g$, with $g \in \{1, 2, ...10\}$ spanned a proportion $.08g$ of the screen width. In Experiment 2, gap $g$ spanned a proportion $.061 + .089g$ of the screen width. The main reason for conducting Experiment 2 was that we found the gaps used in Experiment 1 resulted in low error rates and few sequential effects for the smaller gaps. Other motivations for Experiment 2 will be explained later.

Both experiments were conducted via the web, using a web portal set up for psychology studies. Participants were prescreened for their ability to understand English instructions, and were paid $4 for the 10–15 minutes required to complete the experiment. Two participants in Experiment 1 and one participant in Experiment 2 were excluded from data analysis because their accuracy was below 20%. The portal was opened for long enough to obtain good data from 76 participants in each Experiment. Individuals were allowed to participate in only one of the two experiments.

### 2.2   Results and Discussion of Human Experiments

Figure 1 summarizes the data from Experiments 1 and 2 (top and bottom rows, respectively). All graphs depict the error on a trial, defined as the signed difference $R_t - S_t$ between the current response, $R_t$, and the current stimulus level $S_t$. The left column plots the error on trial $t$ as a function of $S_{t-1}$ (along the abscissa) and $S_t$ (the different colored lines, as specified by the key between the graphs). Pairs of stimulus gaps (e.g., G1 and G2) have been grouped together to simplify the graph.

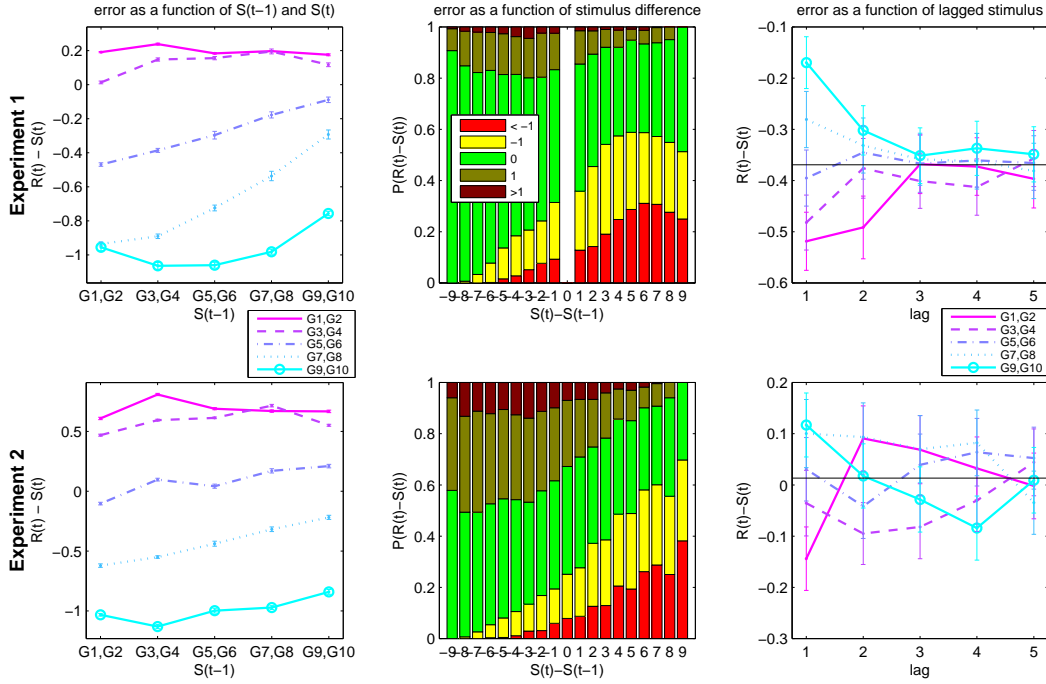

Figure 1: Human data from Experiments 1 (top row) and 2 (bottom row).

The small bars around the point indicate one standard error of the mean. The variation along the abscissa reflects sequential dependencies: assimilation is indicated by pairs of points with positive slopes (larger values of $S_{t-1}$ result in larger $R_t$), and contrast is indicated by negative slopes. The pattern of results across the two experiments is remarkably consistent.

The middle column shows another depiction of sequential dependencies by characterizing the distribution of errors ($R_t - S_t \in \{> 1, 1, 0, -1, < -1\}$) as a function of $S_t - S_{t-1}$. The predominance of assimilative responses is reflected in more $R_t > S_t$ responses when $S_t - S_{t-1} < 0$, and vice-versa. The rightmost column presents the lag profile that characterizes how the stimulus on trial $t - k$ for $k = 1...5$ influences the response on trial $t$. The bars on each point indicate one standard error of the mean. For the purpose of the current work, most relevant is that sequential dependencies in this task may stretch back two or three trials.

## 3    Approaches To Decontamination

From a machine learning perspective, decontamination can be formulated in at least three different ways. First, it could be considered an unsupervised infomax problem of determining a sensation associated with each distinct stimulus such that the sensation sequence has high mutual information with the response sequence. Second, it could be considered a supervised learning problem in which a specialized model is constructed for each individual, using some minimal amount of ground-truth data collected from that individual. Here, the ground truth is the stimulus-sensation correspondence, which can be obtained—in principle, even with unknown stimuli—by laborious data collection techniques, such as asking individuals to provide a full preference ordering or multiple partial orderings over sets of stimuli, or asking individuals to provide multiple ratings of a stimulus in many different contexts, so as to average out sequential effects. Third, decontamination models could be built based on ground-truth data for one group of individuals and then tested on another group. In this paper, we adopt this third formulation of the problem.

Formally, the decontamination problem involves inferring the sequence of (unobserved) sensations given the complete response sequence. To introduce some notation, let $R^p_{t_1, t_2}$ denote the sequence of responses made by participant $p$ on trials $t_1$ through $t_2$ when shown a sequence of stimuli that

evoke the sensation sequence $S_{t_1,t_2}^p$.[1] Decontamination can be cast as computing the expectation or probability over $S_{1,T}^p$ given $R_{1,T}^p$, where $T$ is the total number of judgments made by the individual.

Although psychological theories of human judgment address an altogether different problem—that of predicting $R_t^p$, the response on trial $t$, given $S_{1,t}^p$ and $R_{1,t-1}^p$—they can inspire decontamination techniques. Two classes of psychological theories correspond to two distinct function approximation techniques. Many early models of sequential dependencies, culminating in the work of DeCarlo and Cross (1990), are framed in terms of autoregression. In contrast, other models favor highly flexible, nonlinear approaches that allow for similarity-based assimilation and contrast, and independent representations for each response label (e.g., Petrov & Anderson, 2005). Given the discrete stimuli and responses, a lookup table seems the most general characterization of these models.

We explore a two-dimensional space of decontamination techniques. The first dimension of this space is the model class: regression, lookup table, or an additive hybrid. We define our regression model estimating $S_t$ as:

$$\text{REG}_t(m, n) = \alpha + \boldsymbol{\beta} \cdot R_{t-m+1,t} + \boldsymbol{\gamma} \cdot S_{t-n,t-1}, \tag{1}$$

where the model parameters $\boldsymbol{\beta}$ and $\boldsymbol{\gamma}$ are vectors, and $\alpha$ is a scalar. Similarly, we define our lookup table $\text{LUT}_t(m, n)$ to produce an estimate of $S_t$ by indexing over the $m$ responses $R_{t-m+1,t}$ and the $n$ sensations $S_{t-n,t-1}$. Finally, we define an additive hybrid, $\text{REG} \oplus \text{LUT}(m, n)$ by first constructing a regression model, and then building a lookup table on the residual error, $S_t - \text{REG}_t(m, n)$. The motivation for the hybrid is the complementarity of the two models, the regression model capturing linear regularities and the lookup table representing arbitrary nonlinear relationships.

The second dimension in our space of decontamination techniques specifies how inference is handled. Decontamination is fundamentally a problem of inferring unobserved states. To utilize any of the models above for $n > 0$, sensations $S_{t-n,t-1}$ must be estimated. Although time flows in one direction, inference flows in two: in psychological models, $R_t$ is influenced by both $S_t$ and $S_{t-1}$; this translates to a dependence of $S_t$ on both $S_{t-1}$ and $S_{t+1}$ when conditioned on $R_{1,T}$. To handle inference properly, we construct a linear-chain conditional random field (Lafferty, McCallum, & Pereira, 2001; Sutton & McCallum, 2007). As an alternative to the conditional random field (hereafter, *CRF*), we also consider a *simple* approach in which we simply set $n = 0$ and discard the sensation terms in our regression and lookup tables. At the other extreme, we can assume an *oracle* that provides $S_{t-n,t-1}$; this oracle approach offers an upper bound on achievable performance.

We explore the full Cartesian product of approaches consisting of models chosen from $\{\text{REG}, \text{LUT}, \text{REG} \oplus \text{LUT}\}$ and inference techniques chosen from $\{\text{SIMPLE}, \text{CRF}, \text{ORACLE}\}$. The SIMPLE and ORACLE approaches are straightforward classic statistics, but we need to explain how the different models are incorporated into a CRF. The linear-chain CRF is a distribution

$$P(S_{1,T}|R_{1,T}) = \frac{1}{Z(R_{1,T})} \exp \left\{ \sum_{t=1}^{T} \sum_{k=1}^{K} \lambda_k f_k(t, S_{t-1,t}, R_{1,T}) \right\} \tag{2}$$

with a given set of *feature functions*, $\{f_k\}$. The linear combination of these functions determines the potential at some time $t$, denoted $\Phi_t$, where a higher potential reflects a more likely configuration of variables. To implement a CRF-REG model, we would like the potential to be high when the regression equation is satisfied, e.g., $\Phi_t = -(\text{REG}_t(m, n) - S_t)^2$. Simply expanding this error yields a collection of first and second order terms. Folding the terms not involving the sensations into the normalization constant, the following terms remain for $\text{REG}(2,1)$: $S_t$, $R_t S_t$, $S_t^2$, $R_t S_{t-1}$, $R_{t-1} S_t$, and $S_t S_{t-1}$.[2] The regression potential function can be obtained by making each of these terms into a real-valued feature, and determining the $\lambda$ parameters in Equation 2 to yield the $\alpha$, $\boldsymbol{\beta}$, and $\boldsymbol{\gamma}$ parameters in Equation 1.[3]

The CRF-LUT model could be implemented using indicator features, as is common in CRF models, but this approach yields an explosion of free parameters: a feature would be required for each cell of

the table and each value of $S_t$, yielding $10^4$ free parameters for a gap detection task with a modest CRF-LUT$(2, 1)$. Instead, we opted for the direct analog of the CRF-REG: encouraging configurations in which $S_t$ is consistent with LUT$_t(m, n)$ via potential $\Phi_t = -(\text{LUT}_t(m, n) - S_t)^2$. This approach yields three real-valued features: LUT$_t(m, n)^2$, $S_t{}^2$, and LUT$_t(m, n)S_t$. (Remember that lookup table values are indexed by $S_{t-1}$, and therefore cannot be folded into the normalization constant.)

Finally, the CRF-REG$\oplus$LUT is a straightforward extension of the models we've described, based on the potential $\Phi_t = -(\text{REG}_t(m, n) + \text{LUT}_t(m, n) - S_t)^2$, which still has only quadratic terms in $S_t$ and $S_{t-1}$. Having now described a $3 \times 3$ space of decontamination approaches, we turn to the details of our decontamination experiments.

### 3.1 Debiasing and Decompressing

Although our focus is on decontaminating sequential dependencies, or *desequencing*, the quality of human judgments can be reduced by at least three other factors. First, individuals may have an overall *bias* toward smaller or larger ratings. Second, individuals may show *compression*, possibly nonlinear, of the response range. Third, there may be *slow drift* in the center or spread of the response range, on a relatively long time scale. All of these factors are likely to be caused at least in part by trial-to-trial sequential effects. For example, compression will be a natural consequence of assimilation because the endpoints of the response scale will move toward the center. Nonetheless we find it useful to tease apart the factors that are easy to describe (bias, compression) from those that are more subtle (assimilation, contrast).

In the data from our two experiments, we found no evidence of drift, as determined by the fact that regression models with moving averages of the responses did not improve predictions. This finding is not terribly surprising given that the entire experiment took only 10–15 minutes to complete. We briefly describe how we remove bias and compression from our data. Decompression can be achieved with a LUT$(1, 0)$, which maps each response into the expected sensation. For example, in Experiment 1, the shortest stimuli reported as G1 and G2 with high accuracy, but the longest stimuli tended to be underestimated by all participants. The LUT$(1, 0)$ compensates for this compression by associating responses G8 and G9 with higher sensation levels if the table entries are filled based on the training data according to: LUT$_t(1, 0) \equiv E[S_t | R_t]$. All of the higher order lookup tables, LUT$(m, n)$, for $m \geq 1$ and $n \geq 0$, will also perform nonlinear decompression in the same manner. The REG models alone will also achieve decompression, though only linear decompression.

We found ample evidence of individual biases in the use of the response scale. To *debias* the data, we compute the mean response of a particular participant $p$, $\bar{R}^p \equiv 1/T \sum R_t^p$, and ensure the means are homogeneous via the constraint $R_t^p - \bar{R}^p = S_t^p - \bar{S}^p$. Assuming that the mean sensation is identical for all participants—as it should be in our experiments—debiasing can be incorporated into the lookup tables by storing not $E[S_t | R_t...]$, but rather $E[S_t^p + \bar{R}^p | R_t...]$, and recovering the sensation for a particular individual using LUT$(m, n) - \bar{R}^p$. (This trick is necessary to index into the lookup table with discrete response levels. Simply normalizing individuals' responses will yield noninteger responses.) Debiasing of the regression models can be achieved by adding a $\bar{R}^p$ term to the regression. Note that this extra term—whether in the lookup table retrieval or the regression—results in additional features involving combinations of $\bar{R}^p$ and $S_t$, $S_{t-1}$, and LUT$(m, n)$ being added to the three CRF models.

### 3.2 Modeling Methodology

In all the results we report on, we use a one-back response history, i.e., $m = 2$. Therefore, the SIMPLE models are REG$(2, 0)$, LUT$(2, 0)$, and REG$\oplus$LUT$(2, 0)$, the ORACLE and CRF models are REG$(2, 1)$, LUT$(2, 1)$, and REG$\oplus$LUT$(2, 1)$. In the ORACLE models, $S_{t-1}$ is assumed to be known when $S_t$ is estimated; in the CRF models, the sensations are all inferred. The models are trained via multiple splits of the available data into equal-sized training and test sets (38 participants per set). Parameters of the SIMPLE-REG and ORACLE-REG models are determined by least-squares regression on the training set. Entries in the SIMPLE-LUT and ORACLE-LUT are the expectation over trials and participants: $E[S_t^p + \bar{R}^p | R_t, R_{t-1}, ...]$. The SIMPLE-REG$\oplus$LUT and ORACLE-REG$\oplus$LUT models are trained first by obtaining the regression coefficients, and then filling lookup table entries with the expected residual, $E[S_t^p - \text{REG}_t^p | R_t, R_{t-1}, ...]$. For the CRF models, the feature coefficients $\{\lambda_k\}$ are obtained via gradient descent and the forward-backward algorithm, as detailed in Sutton

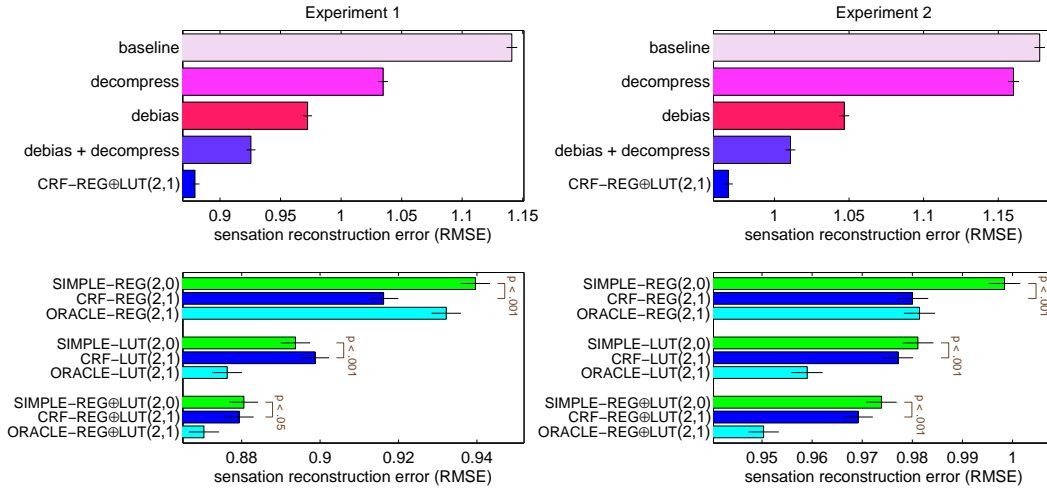

Figure 2: Results from Experiment 1 (left column) and Experiment 2 (right column). The top row compares the reduction in prediction error for different types of decontamination. The bottom row compares reduction in prediction error for different desequencer algorithms.

and McCallum (2007). The lookup tables used in the CRF-LUT and CRF-REG⊕LUT are the same as those in the ORACLE-LUT and ORACLE-REG⊕LUT models. The CRF $\lambda$ parameters are initialized to be consistent with our notion of the potential as the negative squared error, using initialization values obtained from the regression coefficients of the ORACLE-REG model. This initialization is extremely useful because it places the parameters in easy reach of an effective local minimum. No regularization is used on the CRF because of the small number of free parameters (7 for CRF-REG, 5 for CRF-LUT, and 14 for CRF-REG⊕LUT). Each model is used to determine the expected value of $S_t$. We had initially hoped that a Viterbi decoding of the CRF might yield useful predictions, but the expectation proved far superior, most likely because there is not a single path through the CRF that is significantly better than others due to high level of noise in the data.

Beyond the primary set of models described above, we explored several other models. We tested models in which the sensation and/or response values are log transformed, because sensory transduction introduces logarithmic compression. However, these models do not reliably improve decontamination. We examined higher-order regression models, i.e., $m > 2$. These models are helpful for Experiment 1, but only because we inadvertently introduced structure into the sequences via the constraint that each stimulus had to be presented once before it could be repeated. The consequence of this constraint is that a series of small gaps predicted a larger gap on the next trial, and vice-versa. One reason for conducting Experiment 2 was to eliminate this constraint. It also eliminated the benefit of higher-order regression models. We also examined switched regression models whose parameters were contingent on the current response. These models do not significantly outperform the REG⊕LUT models.

## 4   Results

Figure 2 shows the root mean squared error (RMSE) between the ground-truth sensation and the model-estimated sensation over the set of validation subjects for 100 different splits of the data. The left and right columns present results for Experiments 1 and 2, respectively. In the top row of the figure, we compare baseline performance with no decontamination—where the sensation prediction is simply the participant's actual response (pink bar)—against decompression alone (magenta bar), debiasing alone (red bar), debiasing and decompression (purple bar), and the best full decontamination model, which includes debiasing, decompression, and desequencing (blue bar). The difference between each pair of these results is highly reliable, indicating that bias, compression, and recency effects all contribute to the contamination of human judgments.

The reduction of error due to debiasing is 14.8% and 11.1% in Experiments 1 and 2, respectively. The further reduction in error when decompressing is incorporated is 4.8% and 3.4% in Experiments 1 and 2. Finally, the further reduction in error when desequencing is incorporated is 5.0% and 4.1% in Experiments 1 and 2. We reiterate that bias and compression likely have at least part of their basis in sequential dependencies. Indeed models like CRF-REG⊕LUT perform nearly as well even without separate debiasing and decompression corrections.

The bottom row of Figure 2 examines the relative performance of the nine models defined by the Cartesian product of model type (REG, LUT and REG⊕LUT) and inference type (SIMPLE, CRF, and ORACLE). The joint model REG⊕LUT that exploits both the regularity of the regression model and the flexibility of the lookup table clearly works better than either REG or LUT in isolation. Comparing SIMPLE, which ignores the mutual constraints provided by the inferred sensations, to to CRF, which exploits bidirectional temporal constraints, we see that the CRF inference produces reliably better results in five of six cases, as evaluated by paired t-tests. We do not have a good explanation for the advantage of SIMPLE-LUT over CRF-LUT in Experiment 1, although there are some minor differences in how the lookup tables for the two models are constructed, and we are investigating whether those differences might be responsible. We included the ORACLE models to give us a sense of how much improvement we might potentially obtain, and clearly there is still some potential gain as indicated by ORACLE-REG⊕LUT.

## 5 Discussion

Psychologists have long been struck by the relativity of human judgments and have noted that relativity limits how well individuals can communicate their internal sensations, impressions, and evaluations via rating scales. We've shown that decontamination techniques can improve the quality of judgments, reducing error by over 20% Is a 20% reduction significant? In the Netflix competition, if this improvement in the reliability of the available ratings translated to a comparable improvement in the collaborative filtering predictions, it would have been of critical significance.

In this paper, we explored a fairly mundane domain: estimating the gap between pairs of dots on a computer monitor. The advantage of starting our explorations in this domain is that it provided us with ground truth data for training and evaluation of models. Will our conclusions about this sensory domain generalize to more subjective and emotional domains such as movies and art? We are currently designing a study in which we will collect liking judgments for paintings. Using the models we developed for this study, we can obtain a decontamination of the ratings and identify pairs of paintings where the participant's ratings conflict with the decontaminated impressions. Via a later session in which we ask participants for pairwise preferences, we can determine whether the decontaminator or the raw ratings are more reliable. We have reason for optimism because all evidence in the psychological literature suggests that corruption occurs in the mapping of internal states to responses, and there's no reason to suspect that the mapping is different for different types of sensations. Indeed, it seems that if even responses to simple visual stimuli are contaminated, responses to more complex stimuli with a more complex judgment task will be even more vulnerable.

One key limitation of the present work is that it examines unidimensional stimuli, and any interesting domain will involve multidimensional stimuli, such as movies, that could be rated in many different ways depending on the current focus of the evaluator. Anchoring likely determines relevant dimensions as well as the reference points along those dimensions, and it may require a separate analysis to decontaminate this type of anchor.

On the positive side, the domain is ripe for further explorations, and our work suggests many directions for future development. For instance, one might better leverage the CRF's ability to predict not just the expected sensation, but the distribution over sensations. Alternatively, one might pay closer attention to the details of psychological theory in the hope that it provides helpful constraints. One such hint is the finding that systematic effects of sequences have been observed on response latencies in judgment tasks (Lacouture, 1997); therefore, latencies may prove useful for decontamination.

A *Wired Magazine* article on the Netflix competition was entitled, "This psychologist might outsmart the math brains competing for the Netflix prize" (Ellenberg, March 2008). This provocative title didn't turn out to be true, but the title did suggest—consistent with the findings of our research— that the math brains may do well to look inward at the mechanisms of their own brains.

## Acknowledgments

This research was supported by NSF grants BCS-0339103, BCS-720375, and SBE-0518699. The fourth author was supported by an NSF Graduate Student Fellowship. We thank Owen Lewis for conducting initial investigations and discussions that allowed us to better understand the various cognitive models, and Dr. Dan Crumly for the lifesaving advice on numerical optimization techniques.

## Footnotes

[1] We are switching terminology: in the discussion of our experiment, $S$ refers to the stimulus. In the discussion of decontamination, $S$ will refer to the sensation. The difference is minor because the stimulus and sensation are in one-to-one correspondence.

[2] The terms $R_{t-1} S_{t-1}$ and $S_{t-1}^2$ are omitted because they correspond to $R_t S_t$ and $S_t^2$, respectively.

[3] As we explain shortly, the $\{\lambda_k\}$ are determined by CRF training; our point here is that the CRF has the capacity to represent a least-squares regression solution.

## References

DeCarlo, L. T., & Cross, D. V. (1990). Sequential effects in magnitude scaling: Models and theory. *Journal of Experimental Psychology: General*, *119*, 375–396.

Ellenberg, J. (March 2008). This psychologist might outsmart the math brains competing for the netflix prize. *Wired Magazine*, *16*. (http://www.wired.com/techbiz/media/magazine/16-03/mf_netflix?currentPage=all#)

Furnham, A. (1986). The robustness of the recency effect: Studies using legal evidence. *Journal of General Psychology*, *113*, 351–357.

Hogarth, R. M., & Einhorn, H. J. (1992). Order effects in belief updating: The belief adjustment model. *Cognitive Psychology*, *24*, 1–55.

Koren, Y. (August 2009). *The bellkor solution to the netflix grand prize.*

Lacouture, Y. (1997). Bow, range, and sequential effects in absolute identification: A response-time analysis. *Psychological Research*, *60*, 121-133.

Lafferty, J., McCallum, A., & Pereira, F. (2001). Conditional random fields: Probabilistic models for segmenting and labeling sequence data. In *International conference on machine learning* (pp. 282–289). San Mateo, CA: Morgan Kaufmann.

Laming, D. R. J. (1984). The relativity of "absolute" judgements. *Journal of Mathematical and Statistical Psychology*, *37*, 152–183.

Miller, G. A. (1956). The magical number seven, plus or minus two: Some limits on our capacity for information processing. *Psychological Review*, *63*, 81–97.

Mumma, G. H., & Wilson, S. B. (2006). Procedural debiasing of primacy/anchoring effects in clinical-like judgments. *Journal of Clinical Psychology*, *51*, 841–853.

Parducci, A. (1965). Category judgment: A range-frequency model. *Psychological Review*, *72*, 407–418.

Parducci, A. (1968). The relativism of absolute judgment. *Scientific American*, *219*, 84–90.

Petrov, A. A., & Anderson, J. R. (2005). The dynamics of scaling: A memory-based anchor model of category rating and identification. *Psychological Review*, *112*, 383–416.

Stewart, N., Brown, G. D. A., & Chater, N. (2005). Absolute identification by relative judgment. *Psychological Review*, *112*, 881–911.

Sutton, C., & McCallum, A. (2007). An introduction to conditional random fields for relational learning. In L. Getoor & B. Taskar (Eds.), *Introduction to statistical relational learning.* Cambridge, MA: MIT Press.

Tversky, A., & Kahneman, D. (1974). Judgment under uncertainty: Heuristics and biases. *Science*, *185*, 1124–1131.

Wedell, D. H., Parducci, A., & Lane, M. (1990). Reducing the dependence of clinical judgment on the immediate context: Effects of number of categories and type of anchors. *Journal of Personality and Social Psychology*, *58*, 319–329.

